# Designing neurophysiology experiments to optimally constrain receptive field models along parametric submanifolds.

**Jeremy Lewi** *
School of Bioengineering
Georgia Institute of Technology
jeremy@lewi.us

**Robert Butera**
School of Electrical and Computer Engineering
Georgia Institute of Technology
rbutera@ece.gatech.edu

**David M. Schneider**
Departments of Neurobiology and Psychology
Columbia University
dms2159@columbia.edu

**Sarah M. N. Woolley**
Department of Psychology
Columbia University
sw2277@columbia.edu

**Liam Paninski** †
Department of Statistics and Center for Theoretical Neuroscience
Columbia University
liam@stat.columbia.edu

## Abstract

Sequential optimal design methods hold great promise for improving the efficiency of neurophysiology experiments. However, previous methods for optimal experimental design have incorporated only weak prior information about the underlying neural system (e.g., the sparseness or smoothness of the receptive field). Here we describe how to use stronger prior information, in the form of parametric models of the receptive field, in order to construct optimal stimuli and further improve the efficiency of our experiments. For example, if we believe that the receptive field is well-approximated by a Gabor function, then our method constructs stimuli that optimally constrain the Gabor parameters (orientation, spatial frequency, etc.) using as few experimental trials as possible. More generally, we may believe a priori that the receptive field lies near a known sub-manifold of the full parameter space; in this case, our method chooses stimuli in order to reduce the uncertainty along the tangent space of this sub-manifold as rapidly as possible. Applications to simulated and real data indicate that these methods may in many cases improve the experimental efficiency.

## 1 Introduction

A long standing problem in neuroscience has been collecting enough data to robustly estimate the response function of a neuron. One approach to this problem is to sequentially optimize a series of experiments as data is collected [1, 2, 3, 4, 5, 6]. To make optimizing the design tractable, we typically need to assume our knowledge has some nice mathematical representation. This restriction often makes it difficult to include the types of prior beliefs held by neurophysiologists; for example that the receptive field has some parametric form such as a Gabor function [7]. Here we consider

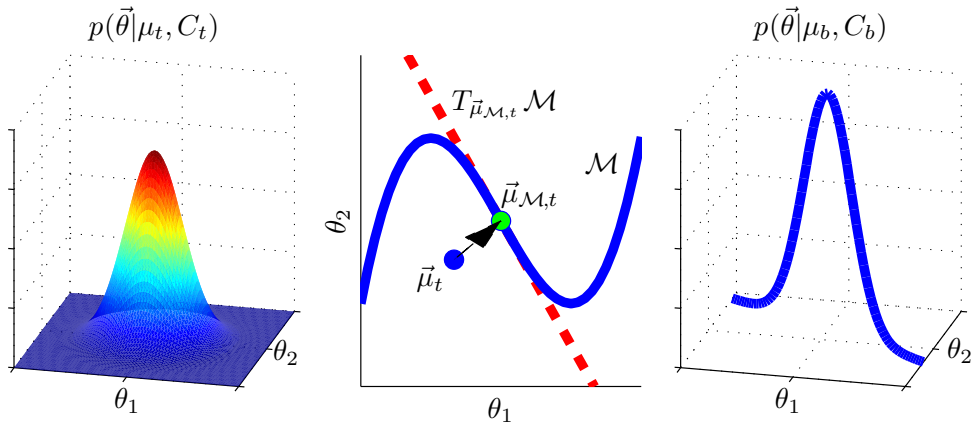

Figure 1: A schematic illustrating how we use the manifold to improve stimulus design. Our method begins with a Gaussian approximation of the posterior on the full model space after $t$ trials, $p(\vec{\theta}|\vec{\mu}_t, \boldsymbol{C}_t)$. The left panel shows an example of this Gaussian distribution when $dim(\vec{\theta}) = 2$. The next step involves constructing the tangent space approximation of the manifold $\mathcal{M}$ on which $\vec{\theta}$ is believed to lie, as illustrated in the middle plot; $\mathcal{M}$ is indicated in blue. The MAP estimate (blue dot) is projected onto the manifold to obtain $\vec{\mu}_{\mathcal{M},t}$ (green dot). We then compute the tangent space (dashed red line) by taking the derivative of the manifold at $\vec{\mu}_{\mathcal{M},t}$. The tangent space is the space spanned by vectors in the direction parallel to $\mathcal{M}$ at $\vec{\mu}_{\mathcal{M},t}$. By definition, in the neighborhood of $\vec{\mu}_{\mathcal{M},t}$, moving along the manifold is roughly equivalent to moving along the tangent space. Thus, the tangent space provides a good local approximation of $\mathcal{M}$. In the right panel we compute $p(\vec{\theta}|\vec{\mu}_{b,t}, C_{b,t})$ by evaluating $p(\vec{\theta}|\vec{\mu}_t, \boldsymbol{C}_t)$ on the tangent space. The resulting distribution concentrates its mass on models which are probable under $p(\vec{\theta}|\vec{\mu}_t, \boldsymbol{C}_t)$ and close to the manifold.

the problem of incorporating this strong prior knowledge into an existing algorithm for optimizing neurophysiology experiments [8].

We start by assuming that a neuron can be modeled as a generalized linear model (GLM). Our prior knowledge defines a subset of all GLMs in which we expect to find the best model of the neuron. We represent this class as a sub-manifold in the parameter space of the GLM. We use the manifold to design an experiment which will provide the largest reduction in our uncertainty about the unknown parameters. To make the computations tractable we approximate the manifold using the tangent space evaluated at the maximum a posteriori (MAP) estimate of the parameters projected onto the manifold. Despite this rather crude approximation of the geometry of the manifold, our simulations show that this method can significantly improve the informativeness of our experiments. Furthermore, these methods work robustly even if the best model does not happen to lie directly on the manifold.

## 2   Methods

We begin by summarizing the three key elements of an existing algorithm for optimizing neurophysiology experiments. A more thorough discussion is available in [8]. We model the neuron's response function as a mapping between the neuron's input at time $t$, $\vec{s}_t$, and its response, $r_t$. We define the input rather generally as a vector which may consist of terms corresponding to a stimulus, e.g. an image or a sound, or the past activity of the neuron itself, $\{r_{t-1}, r_{t-2}, \ldots\}$. The response, $r_t$, is typically a non-negative integer corresponding to the number of spikes observed in a small time window. Since neural responses are typically noisy, we represent the response function as a conditional distribution, $p(r_t|\vec{s}_t, \vec{\theta})$. In this context, optimizing the experimental design means picking the input for which observing the response will provide the most information about the parameters $\vec{\theta}$ defining the conditional response function.

The first important component of this algorithm is the assumption that $p(r_t|\vec{s}_t, \vec{\theta})$ can be adequately approximated by a generalized linear model [9, 10]. The likelihood of the response depends on the firing rate, $\lambda_t$, which is a function of the input,

$$\lambda_t = E(r_t) = f\left(\vec{\theta}^T \vec{s}_t\right),\tag{1}$$

where $f()$ is some nonlinear function which is assumed known[1]. To identify the response function, we need to estimate the coefficients of the linear projection, $\vec{\theta}$. One important property of the GLM is that we can easily derive sufficient conditions to ensure the log-likelihood is concave [11].

The second key component of the algorithm is that we may reasonably approximate the posterior on $\vec{\theta}$ as Gaussian. This approximation is justified by the log-concavity of the likelihood function and asymptotic normality of the posterior distribution given sufficient data [12]. As a result, we can recursively compute a Gaussian approximation of the full posterior, $p(\vec{\theta}|\boldsymbol{r_{1:t}}, \boldsymbol{s_{1:t}}) \approx p(\vec{\theta}|\vec{\mu}_t, \boldsymbol{C}_t)$ [8]. Here $(\vec{\mu}_t, \boldsymbol{C}_t)$ denote the mean and covariance matrix of our Gaussian approximation: $\vec{\mu}_t$ is set to the MAP estimate of $\vec{\theta}$, and $\boldsymbol{C}_t$ to the inverse Hessian of the log-posterior at $\vec{\mu}_t$.

The final component is an efficient method for picking the optimal input on the next trial, $\vec{s}_{t+1}$. Since the purpose of an experiment is to identify the best model, we optimize the design by maximizing the conditional mutual information between $r_{t+1}$ and $\vec{\theta}$ given $\vec{s}_{t+1}$, $I(\vec{\theta}; r_{t+1}|\vec{s}_{t+1})$. The mutual information measures how much we expect observing the response to $\vec{s}_{t+1}$ will reduce our uncertainty about $\vec{\theta}$. We pick the optimal input by maximizing the mutual information with respect to $\vec{s}_{t+1}$; as discussed in [8], this step, along with the updating of the posterior mean and covariance $(\vec{\mu}_t, \boldsymbol{C}_t)$, may be computed efficiently enough for real-time implementation in many cases.

## 2.1 Optimizing experiments to reduce uncertainty along parameter sub-manifolds.

For the computation of the mutual information to be tractable, the space of candidate models, $\Theta$, must have some convenient form so that we can derive a suitable expression for the mutual information. Intuitively, to select the optimal design, we need to consider how much information an experiment provides about each possible model. Evaluating the mutual information entails an integral over model space, $\Theta$. The problem with incorporating prior knowledge is that if we restrict the model to some complicated subset of model space we will no longer be able to efficiently integrate over the set of candidate models. We address this problem by showing how local geometric approximations to the parameter sub-manifold can be used to guide optimal sampling while still maintaining a flexible, tractable representation of the posterior distribution on the full model space.

In many experiments, neurophysiologists expect a-priori that the receptive field of a neuron will have some low-dimensional parametric structure; e.g the receptive field might be well-approximated by a Gabor function [13], or by a difference of Gaussians [14], or by a low rank spatiotemporal matrix [15, 13]. We can think of this structure as defining a sub-manifold, $\mathcal{M}$, of the full model space, $\Theta$,

$$\mathcal{M} = \{\vec{\theta} : \vec{\theta} = \Psi(\vec{\phi}), \forall \vec{\phi}\}.\tag{2}$$

The vector, $\vec{\phi}$, essentially enumerates the points on the manifold and $\Psi()$ is a function which maps these points into $\Theta$ space. A natural example is the case where we wish to enforce the constraint that $\vec{\theta}$ has some parametric form, e.g. a Gabor function. The basic idea is that we want to run experiments which can identify exactly where on the manifold the optimal model lies.

Since $\mathcal{M}$ can have some arbitrary nonlinear shape, computing the informativeness of a stimulus using just the models on the manifold is not easy. Furthermore, if we completely restrict our attention to models in $\mathcal{M}$ then we ignore the possibility that our prior knowledge is incorrect. Hence, we do not force the posterior distribution of $\vec{\theta}$ to only have support on the manifold. Rather, we maintain a Gaussian approximation of the posterior on the full space, $\Theta$. However, when optimizing our stimuli we combine our posterior with our knowledge of $\mathcal{M}$ in order to do a better job of maximizing the informativeness of each experiment.

Computing the mutual information $I(r_{t+1}; \vec{\theta} | \vec{s}_{t+1}, \boldsymbol{s_{1:t}}, \boldsymbol{r_{1:t}})$ entails an integral over model space weighted by the posterior probability on each model. We integrate over model space because the informativeness of an experiment clearly depends on what we already know (i.e. the likelihood we assign to each model given the data and our prior knowledge). Furthermore, the informativeness of an experiment will depend on the outcome. Hence, we use what we know about the neuron to make predictions about the experimental outcome. Unfortunately, since the manifold in general has some arbitrary nonlinear shape we cannot easily compute integrals over the manifold. Furthermore, we do not want to continue to restrict ourselves to models on the manifold if the data indicates our prior knowledge is wrong.

We can solve both problems by making use of the tangent space of the manifold, as illustrated in Figure 1 [16]. The tangent space is a linear space which provides a local approximation of the manifold. Since the tangent space is a linear subspace of $\Theta$, integrating over $\vec{\theta}$ in the tangent space is much easier than integrating over all $\vec{\theta}$ on the manifold; in fact, the methods introduced in [8] may be applied directly to this case. The tangent space is a local linear approximation evaluated at a particular point, $\vec{\mu}_{\mathcal{M},t}$, on the manifold. For $\vec{\mu}_{\mathcal{M},t}$ we use the projection of $\vec{\mu}_t$ onto the manifold (i.e., $\vec{\mu}_{\mathcal{M},t}$ is the closest point in $\mathcal{M}$ to $\vec{\mu}_t$). Depending on the manifold, computing $\vec{\mu}_{\mathcal{M},t}$ can be nontrivial; the examples considered in this paper, however, all have tractable numerical solutions to this problem.

The challenge is representing the set of models close to $\vec{\mu}_{\mathcal{M},t}$ in a way that makes integrating over the models tractable. To find models on the manifold close to $\vec{\mu}_{\mathcal{M},t}$ we want to perturb the parameters $\vec{\phi}$ about the values corresponding to $\vec{\mu}_{\mathcal{M},t}$. Since $\Psi$ is in general nonlinear, there is no simple expression for the combination of all such perturbations. However, we can easily approximate the set of $\vec{\theta}$ resulting from these perturbations by taking linear combinations of the partial derivatives of $\Psi$ with respect to $\vec{\phi}$. The partial derivative is the direction in $\Theta$ in which $\vec{\theta}$ moves if we perturb one of the manifold's parameters. Thus, the subspace formed by linear combinations of the partial derivatives approximates the set of models on the manifold close to $\vec{\mu}_{\mathcal{M},t}$. This subspace is the tangent space,

$$T_{\vec{\mu}_{\mathcal{M},t}}\mathcal{M} = \{\vec{\theta} : \vec{\theta} = \vec{\mu}_{\mathcal{M},t} + \boldsymbol{B}\vec{b}, \ \forall \vec{b} \in \mathcal{R}^{\dim(\mathcal{M})}\} \qquad \boldsymbol{B} = orth\left(\left[\frac{\partial \Psi}{\partial \phi_1} \cdots \frac{\partial \Psi}{\partial \phi_d}\right]\right), \qquad (3)$$

where $orth$ is an orthonormal basis for the column space of its argument. Here $T_x\mathcal{M}$ denotes the tangent space at the point $x$. The columns of $\boldsymbol{B}$ denote the direction in which $\vec{\theta}$ changes if we perturb one of the manifold's parameters. (In general, the directions corresponding to changes in different parameters are not independent; to avoid this redundancy we compute a set of basis vectors for the space spanned by the partial derivatives.)

We now use our Gaussian posterior on the full parameter space to compute the posterior likelihood of the models in the tangent space. Since the tangent space is a subspace of $\Theta$, restricting our Gaussian approximation, $p(\vec{\theta} | \vec{\mu}_t, \boldsymbol{C}_t)$, to the tangent space means we are taking a slice through our Gaussian approximation of the posterior. Mathematically, we are conditioning on $\vec{\theta} \in T_{\vec{\mu}_{\mathcal{M},t}}\mathcal{M}$. The result is a Gaussian distribution on the tangent space whose parameters may be obtained using the standard Gaussian conditioning formula:

$$p_{tan}(\vec{\theta}|\vec{\mu}_{b,t}, C_{b,t}) = \begin{cases} \mathcal{N}(\vec{b}; \vec{\mu}_{b,t}, C_{b,t}) & if \quad \exists \ \vec{b} \ s.t \ \vec{\theta} = \vec{\mu}_{\mathcal{M},t} + \boldsymbol{B}\vec{b} \\ 0 & if \quad \quad \vec{\theta} \notin T_{\vec{\mu}_{\mathcal{M},t}} \end{cases} \qquad (4)$$

$$\vec{\mu}_{b,t} = -C_{b,t}\boldsymbol{B}^T \boldsymbol{C}_t^{-1}(\vec{\mu}_{\mathcal{M},t} - \vec{\mu}_t) \qquad C_{b,t} = (\boldsymbol{B}^T \boldsymbol{C}_t^{-1}\boldsymbol{B})^{-1} \qquad (5)$$

where $\mathcal{N}$ denotes a normal distribution with the specified parameters. Now, rather than optimizing the stimulus by trying to squeeze the uncertainty $p(\vec{\theta}|\boldsymbol{r_{1:t}}, \boldsymbol{s_{1:t}}, \mathcal{M})$ on the nonlinear manifold $\mathcal{M}$ down as much as possible (a very difficult task in general), we pick the stimulus which best reduces the uncertainty $p_{tan}(\vec{\theta}|\vec{\mu}_{b,t}, C_{b,t})$ on the vector space $T_{\vec{\mu}_{\mathcal{M},t}}$. We can solve this latter problem directly using the methods presented in [8]. Finally, to handle the possibility that $\vec{\theta} \notin \mathcal{M}$, every so often we optimize the stimulus using the full posterior $p(\vec{\theta}|\vec{\mu}_t, \boldsymbol{C}_t)$. This simple modification ensures that asymptotically we do not ignore directions orthogonal to the manifold; i.e., that we do not

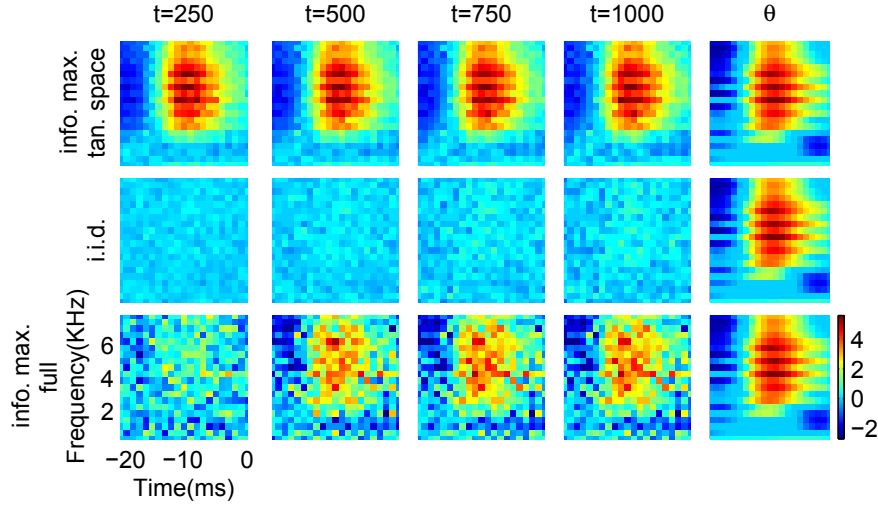

Figure 2: MAP estimates of a STRF obtained using three designs: the new info. max. tangent space design described in the text; an i.i.d. design; and an info. max. design which did not use the assumption that $\vec{\theta}$ corresponds to a low rank STRF. In each case, stimuli were chosen under the spherical power contraint, $||\vec{s}_t||_2 = c$. The true STRF (fit to real zebrafinch auditory responses and then used to simulate the observed data) is shown in the last column. (For convenience we rescaled the coefficients to be between -4 and 4). We see that using the tangent space to optimize the design leads to much faster convergence to the true parameters; in addition, either infomax design significantly outperforms the iid design here. In this case the true STRF did not in fact lie on the manifold $\mathcal{M}$ (chosen to be the set of rank-2 matrices here); thus, these results also show that our knowledge of $\mathcal{M}$ does not need to be exact in order to improve the experimental design.

get stuck obsessively sampling along the incorrect manifold. As a result, $\mu_t$ will always converge asymptotically to the true parameters, even when $\theta \notin M$.

To summarize, our method proceeds as follows:

0. Initial conditions: start with a log-concave (approximately Gaussian) posterior given $t$ previous trials, summarized by the posterior mean, $\vec{\mu}_t$ and covariance, $C_t$.

1. Compute $\vec{\mu}_{\mathcal{M},t}$, the projection of $\vec{\mu}_t$ on the manifold. (The procedure for computing $\vec{\mu}_{\mathcal{M},t}$ depends on the manifold.)

2. Compute the tangent space of $\mathcal{M}$ at $\vec{\mu}_{\mathcal{M},t}$ using Eqn. 3.

3. Compute the posterior restricted to the tangent space, $p_{tan}(\vec{\theta}|\vec{\mu}_{b,t}, C_{b,t})$, using the standard Gaussian conditioning formula (Eqn. 5).

4. Apply the methods in [8] to find the optimal $t+1$ stimulus, and observe the response $r_{t+1}$.

5. Update the posterior by recursively updating the posterior mean and covariance: $\vec{\mu}_t \rightarrow \vec{\mu}_{t+1}$ and $C_t \rightarrow C_{t+1}$ (again, as in [8]), and return to step 1.

## 3 Results

### 3.1 Low rank models

To test our methods in a realistic, high-dimensional setting, we simulated a typical auditory neurophysiology [17, 15, 18] experiment. Here, the objective is to to identify the spectro-temporal receptive field (STRF) of the neuron. The input and receptive field of the neuron are usually represented in the frequency domain because the cochlea is known to perform a frequency decomposition of sound. The STRF, $\theta(\tau, \omega)$, is a 2-d filter which relates the firing rate at time $t$ to the amount of

energy at frequency $\omega$ and time $t - \tau$ in the stimulus. To incorporate this spectrotemporal model in the standard GLM setting, we simply vectorize the matrix $\theta(\tau, \omega)$.

Estimating the STRF can be quite difficult due to its high dimensionality. Several researchers, however, have shown that low-rank assumptions can be used to produce accurate approximations of the receptive field while significantly reducing the number of unknown parameters [19, 13, 15, 20]. A low rank assumption is a more general version of the space-time separable assumption that is often used when studying visual receptive fields [21]. Mathematically, a low-rank assumption means that the matrix corresponding to the STRF can be written as a sum of rank one matrices,

$$\boldsymbol{\Theta} = Mat\,\vec{\theta} = UV^T \tag{6}$$

where $Mat$ indicates the matrix formed by reshaping the vector $\vec{\theta}$ to form the STRF. $U$ and $V$ are low-rank matrices with orthonormal columns. The columns of $U$ and $V$ are the principal components of the column and row spaces of $\boldsymbol{\Theta}$ respectively, and encode the spectral and temporal properties of the STRF, respectively.

We simulated an auditory experiment using an STRF fitted to the actual response of a neuron in the Mesencephalicus lateralis pars dorsalis (MLd) of an adult male zebra finch [18]. To reduce the dimensionality we sub-sampled the STRF in the frequency domain and shortened it in the time domain to yield a $20 \times 21$ STRF. We generated synthetic data by sampling a Poisson process whose instantaneous firing rate was set to the output of a GLM with exponential nonlinearity and $\vec{\theta}$ proportional to the true measured zebra finch STRF.

For the manifold we used the set of $\vec{\theta}$ corresponding to rank-2 matrices. For the STRF we used, the rank-2 assumption turns out to be rather accurate. We also considered manifolds of rank-1 and rank-5 matrices (data not shown), but rank-2 did slightly better. The manifold of rank $r$ matrices is convenient because we can easily project any $\vec{\theta}$ onto $\mathcal{M}$ by reshaping $\vec{\theta}$ as a matrix and then computing its singular-value-decomposition (SVD). $\vec{\mu}_{\mathcal{M},t}$ is the matrix formed by the first $r$ singular vectors of $\vec{\mu}_t$. To compute the tangent space, Eqn. 3, we compute the derivative of $\vec{\theta}$ with respect to each component of the matrices $U$ and $V$. Using these derivatives we can linearly approximate the effect on $\boldsymbol{\Theta}$ of perturbing the parameters of its principal components.

In Figure 3.1 we compare the effectiveness of different experimental designs by plotting the MAP estimate $\vec{\mu}_t$ on several trials. The results clearly show that using the tangent space to design the experiments leads to much faster convergence to the true parameters. Furthermore, using the assumption that the STRF is rank-2 is beneficial even though the true STRF here is not in fact rank-2.

## 3.2 Real birdsong data

We also tested our method by using it to reshuffle the data collected during an actual experiment to find an ordering which provided a faster decrease in the error of the fitted model. During the experiments, we recorded the responses of MLd neurons when the songs of other birds and ripple noise were presented to the bird (again, as previously described in [18]). We compared a design which randomly shuffled the trials to a design which used our info. max. algorithm to select the order in which the trials are processed. We then evaluated the fitted model by computing the expected log-likelihood of the spike trains, $\sum_\tau E_{\vec{\theta}|\vec{\mu}_t, C_t} \log p(r_\tau|\vec{s}_\tau, \vec{\theta})$. $\tau$ denotes all the observations made when inputs in a test set are played to the bird.

To constrain the models we assume the STRF is low-rank and that its principal components are smooth. The smoothing prior means that if we take the Fourier transform of the principal components, the Fourier coefficients of high frequencies should be zero with high probability. In other words, each principal component (the columns of $U$ and $V$) should be a linear combination of sinusoidal functions with low frequencies. In this case we can write the STRF as

$$\boldsymbol{\Theta} = \mathcal{F}\nu\boldsymbol{\omega}\eta^T\boldsymbol{\mathcal{T}}^T. \tag{7}$$

Each column of $\mathcal{F}$ and $\mathcal{T}$ is a sine or cosine function representing one of the basis functions of the principal spectral (columns of $\mathcal{F}$) or temporal (columns of $\mathcal{T}$) components of the STRF. Each column of $\nu$ and $\eta$ determines how we form one of the principal components by combining sine and cosine functions. $\boldsymbol{\omega}$ is a diagonal matrix which specifies the projection of $\boldsymbol{\Theta}$ onto each principal

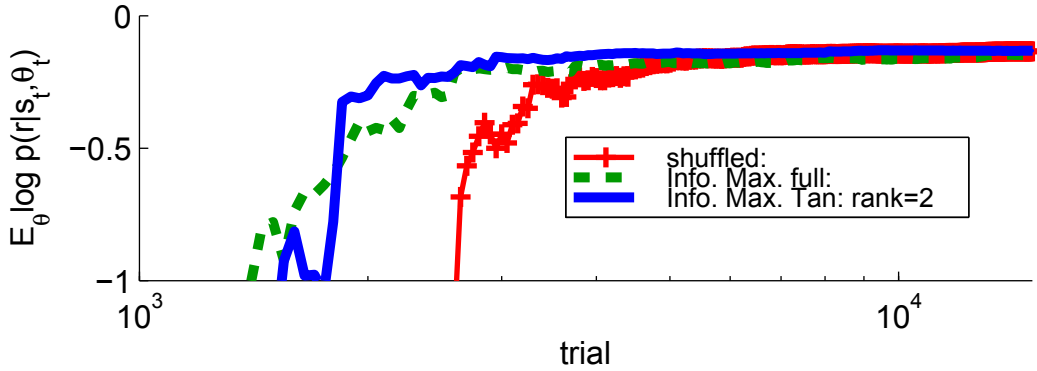

Figure 3: Plots comparing the performance of an info. max. design, an info. max. design which uses the tangent space, and a shuffled design. The manifold was the set of rank 2 matrices. The plot shows the expected log-likelihood (prediction accuracy) of the spike trains in response to a birdsong in the test set. Using a rank 2 manifold to constrain the model produces slightly better fits of the data.

component. The unknown parameters in this case are the matrices $\nu$, $\eta$, and $\boldsymbol{\omega}$. The sinusoidal functions corresponding to the columns of $\mathcal{F}$ and $\mathcal{T}$ should have frequencies $\{0, \ldots, f_{o,f}m_f\}$ and $\{0, \ldots, f_{o,t}m_t\}$ respectively. $f_{o,f}$ and $f_{o,t}$ are the fundamental frequencies and are set so that 1 period corresponds to the dimensions of the STRF. $m_f$ and $m_t$ are the largest integers such that $f_{o,f}m_f$ and $f_{o,t}m_t$ are less than the Nyquist frequency. Now to enforce a smoothing prior we can simply restrict the columns of $\mathcal{F}$ and $\mathcal{T}$ to sinusoids with low frequencies. To project $\boldsymbol{\Theta}$ onto the manifold we simply need to compute $\nu, \boldsymbol{\omega}$ and $\eta$ by evaluating the SVD of $\mathcal{F}^T \boldsymbol{\Theta} \mathcal{T}$.

The results, Figure 3, show that both info. max. designs significantly outperform the randomly shuffled design. Furthermore, incorporating the low-rank assumption using the tangent space improves the info. max. design, albeit only slightly; the estimated STRF's are shown in Figure 4. It is worth noting that in an actual online experiment, we would expect a larger improvement with the info. max. design, since during the experiment we would be free to pick any input. Thus, the different designs could choose radically different stimulus sets; in contrast, when re-analyzing the data offline, all we can do is reshuffle the trials, but the stimulus sets remain the same in the info. max. and iid settings here.

## 4 Conclusion

We have provided a method for incorporating detailed prior information in existing algorithms for the information-theoretic optimal design of neurophysiology experiments. These methods use realistic assumptions about the neuron's response function and choose significantly more informative stimuli, leading to faster convergence to the true response function using fewer experimental trials. We expect that the inclusion of this strong prior information will help experimentalists contend with the high dimensionality of neural response functions.

## 5 Acknowledgments

We thank Vincent Vu and Bin Yu for helpful conversations. JL is supported by the Computational Science Graduate Fellowship Program administered by the DOE under contract DE-FG02-97ER25308 and by the NSF IGERT Program in Hybrid Neural Microsystems at Georgia Tech via grant number DGE-0333411. LP is supported by an NSF CAREER award and a Gatsby Initiative in Brain Circuitry Pilot Grant.

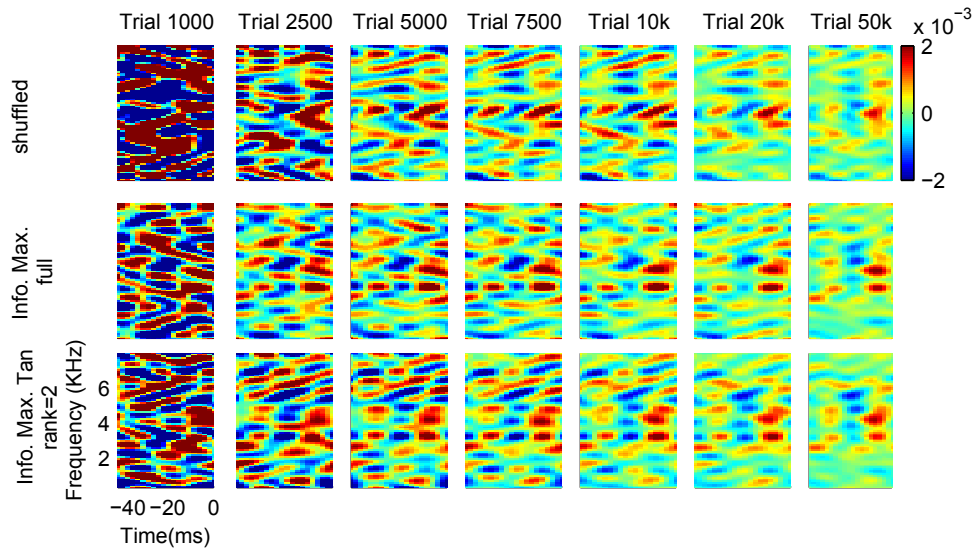

Figure 4: The STRFs estimated using the bird song data. We plot $\vec{\mu}_t$ for trials in the interval over which the expected log-likelihood of the different designs differed the most in Fig. 3. The info. max. designs converge slightly faster than the shuffled design. In these results, we smoothed the STRF by only using frequencies less than or equal to $10 f_{o,f}$ and $2 f_{o,t}$.

## Footnotes

*http://www.lewilab.org

†http://www.stat.columbia.edu/∼liam/

[1]It is worth noting that this simple GLM can be generalized in a number of directions; we may include spike-history effects, nonlinear input terms, and so on [10].

## References

[1] P. Foldiak, *Neurocomputing* **38–40**, 1217 (2001).

[2] R. C. deCharms, *et al.*, *Science* **280**, 1439 (1998).

[3] T. Gollisch, *et al.*, *Journal of Neuroscience* **22**, 10434 (2002).

[4] F. Edin, *et al.*, *Journal of Computational Neuroscience* **17**, 47 (2004).

[5] C. Machens, *et al.*, *Neuron* **47**, 447 (2005).

[6] K. N. O'Connor, *et al.*, *Journal of Neurophysiology* **94**, 4051 (2005).

[7] D. L. Ringach, *J Neurophysiol* **88**, 455 (2002).

[8] J. Lewi, *et al.*, *Neural Computation* **21** (2009).

[9] E. Simoncelli, *et al.*, *The Cognitive Neurosciences*, M. Gazzaniga, ed. (MIT Press, 2004).

[10] L. Paninski, *et al.*, *Computational Neuroscience: Theoretical Insights into Brain Function* (Elsevier, 2007), chap. Statistical models for neural encoding, decoding, and optimal stimulus design.

[11] L. Paninski, *Network: Computation in Neural Systems* **15**, 243 (2004).

[12] L. Paninski, *Neural Computation* **17**, 1480 (2005).

[13] A. Qiu, *et al.*, *J Neurophysiol* **90**, 456 (2003).

[14] C. Enroth-Cugell, *et al.*, *Journal of Physiology* **187**, 517 (1966).

[15] J. F. Linden, *et al.*, *Journal of Neurophysiology* **90**, 2660 (2003).

[16] J. M. Lee, *Introduction to Smooth Manifolds* (Springer, 2000).

[17] F. E. Theunissen, *et al.*, *Journal of Neuroscience* **20**, 2315 (2000).

[18] S. M. Woolley, *et al.*, *The Journal of Neuroscience* **26**, 2499 (2006).

[19] D. A. Depireux, *et al.*, *Journal of Neurophysiology* **85**, 1220 (2001).

[20] M. B. Ahrens, *et al.*, *Network* **19**, 35 (2008).

[21] G. C. DeAngelis, *et al.*, *J Neurophysiol* **69**, 1091 (1993).

